# A Model of the Phonological Loop: Generalization and Binding

**Randall C. O'Reilly**
Department of Psychology
University of Colorado Boulder
345 UCB
Boulder, CO 80309
oreilly@psych.colorado.edu

**Rodolfo Soto**
Department of Psychology
University of Colorado Boulder
345 UCB
Boulder, CO 80309

## Abstract

We present a neural network model that shows how the prefrontal cortex, interacting with the basal ganglia, can maintain a sequence of phonological information in activation-based working memory (i.e., the *phonological loop*). The primary function of this phonological loop may be to transiently encode arbitrary bindings of information necessary for tasks — the combinatorial expressive power of language enables very flexible binding of essentially arbitrary pieces of information. Our model takes advantage of the closed-class nature of phonemes, which allows different neural representations of all possible phonemes at each sequential position to be encoded. To make this work, we suggest that the basal ganglia provide a region-specific update signal that allocates phonemes to the appropriate sequential coding slot. To demonstrate that flexible, arbitrary binding of novel sequences can be supported by this mechanism, we show that the model can generalize to novel sequences after moderate amounts of training.

## 1 Introduction

Sequential binding is a version of the binding problem requiring that the identity of an item and its position within a sequence be bound. For example, to encode a phone number (e.g., 492-0054), one must remember not only the digits, but their order within the sequence. It has been suggested that the brain may have developed a specialized system for this form of binding in the domain of phonological sequences, in the form of the *phonological loop* (Baddeley, 1986; Baddeley, Gathercole, & Papagno, 1998; Burgess & Hitch, 1999). The phonological loop is generally conceived of as a system that can quickly encode a sequence of phonemes and then repeat this sequence back repeatedly. Standard estimates place the capacity of this loop at about 2.5 seconds of "inner speech," and it is widely regarded as depending on the prefrontal cortex (e.g., Paulesu, Frith, & Frackowiak, 1993). We have developed a model of the phonological loop based on our existing framework for understanding how the prefrontal cortex and basal ganglia interact to support

activation-based working memory (Frank, Loughry, & O'Reilly, 2001). This model performs binding by using different neural substrates for the different sequential positions of phonemes. This is a viable solution for a small, closed-class set of items like phonemes. However, through the combinatorial power of language, these phonological sequences can represent a huge number of distinct combinations of concepts. Therefore, this basic maintenance mechanism can be leveraged in many different circumstances to bind information needed for immediate use (e.g., in *working memory* tasks).

A good example of this form of transient, phonologically-dependent binding comes from a task studied by Miyake and Soto (in preparation). In this task, participants saw sequentially-presented colored letters one at a time on a computer display, and had to respond to *targets* of a red X or a green Y, but not to any other color-letter combination (e.g., green X's and red Y's, which were also presented). After an initial series of trials with this set of targets, the targets were switched to be a green X and a red Y. Thus, the task clearly requires binding of color and letter information, and updating of these bindings after the switch condition. Miyake and Soto (in preparation) found that if they simply had participants repeat the word "the" over and over during the task (i.e., *articulatory suppression*), it interfered significantly with performance. In contrast, performing a similar repeated motor response that did not involve the phonological system (repeated foot tapping) did not interfere (but this task did interfere at the same level as articulatory suppression in a control visual search task, so one cannot argue that the interference was simply a matter of differential task difficulty). Miyake and Soto (in preparation) interpret this pattern of results as showing that the phonological loop supports the binding of stimulus features (e.g., participants repeatedly say to themselves "red X, green Y...", which is supported by debriefing reports), and that the use of this phonological system for unrelated information during articulatory suppression leads to the observed performance deficits.

This form of phonological binding can be contrasted with other forms of binding that can be used in other situations and subserved by other brain areas besides the prefrontal cortex. O'Reilly, Busby, and Soto (in press) identify two other important binding mechanisms and their neural substrates in addition to the phonological loop mechanism:

- *Cortical coarse-coded conjunctive binding:* This is where each neural unit codes in a graded fashion for a large number of relatively low-order conjunctions, and many such units are used to represent any given input (e.g., Wickelgren, 1969; Mel & Fiser, 2000; O'Reilly & Busby, 2002). This form of binding takes place within the basic representations in the network that are shaped by gradual learning processes and provides a long-lasting (non-transient) form of binding. In short, these kinds of distributed representations avoid the binding problem in the first place by ensuring that relevant conjunctions are encoded, instead of representing different features using entirely separate, localist units (which is what gives rise to binding problems in the first place). However, this form of binding cannot rapidly encode novel bindings required for specific tasks — the phonological loop mechanism can thus complement the basic cortical mechanism by providing flexible, transient bindings on an ad-hoc basis.

- *Hippocampal episodic conjunctive binding:* Many theories of hippocampal function converge on the idea that it binds together individual elements of an experience into a unitary representation, which can for example be later recalled from partial cues (see O'Reilly & Rudy, 2001 for a review). These hippocampal conjunctive representations are higher-order and more spe-

cific than the lower-order coarse-coded cortical conjunctive representations (i.e., a hippocampal conjunction encodes the combination of many feature elements, while a cortical conjunction encodes relatively few). Thus, the hippocampus can be seen as a specialized system for doing long-term binding of specific episodes, complementing the more generalized conjunctive binding performed by the cortex. Importantly, the hippocampus can also encode these conjunctions rapidly, and therefore it shares some of the same functionality as the phonological loop mechanism (i.e., rapidly encoding arbitrary conjunctions required for tasks). Thus, it is likely that the hippocampus and the prefrontal-mediated working memory system (including the phonological loop) are partially redundant with each other, and work together in many tasks (Cohen & O'Reilly, 1996).

## 2 Prefrontal Cortex and Basal Ganglia in Working Memory

Our model of the phonological loop takes advantage of recent work showing how the prefrontal cortex and basal ganglia can interact to support activation-based working memory (Frank et al., 2001). The critical principles behind this work are as follows:

- Prefrontal cortex (PFC) is specialized relative to the posterior cortex for *robust and rapidly updatable* maintenance of information in an active state (i.e., via persistent firing of neurons). Thus, PFC can quickly update to maintain new information (in this case, the one exposure to a sequence of phonemes), while being able to also protect maintained information from interference from ongoing processing (see O'Reilly, Braver, & Cohen, 1999; Cohen, Braver, & O'Reilly, 1996; Miller & Cohen, 2001 for elaborations and reviews of relevant data).

- Robust maintenance and rapid updating are in fundamental conflict, and require a *dynamic gating mechanism* that can switch between these two modes of operation (O'Reilly et al., 1999; Cohen et al., 1996).

- The basal ganglia (BG) can provide this dynamic gating mechanism via modulatory, disinhibitory connectivity with the PFC. Furthermore, this BG-based gating mechanism provides *selectivity*, such that separate regions of the PFC can be independently updated or allowed to perform robust maintenance. A possible anatomical substrate for these separably updatable PFC regions are the *stripe* structures identified by Levitt, Lewis, Yoshioka, and Lund (1993).

- Active maintenance in the PFC is implemented via a combination of recurrent excitatory connections and intracellular excitatory ionic conductances. This allows the PFC units to generally reflect the current inputs, except when these units have their intracellular maintenance currents activated, which causes them to reflect previously maintained information. See Frank et al. (2001) for more details on the importance of this mechanism.

## 3 Phonological Loop Model

The above mechanisms motivated our modeling of the phonological loop as follows (see Figure 1). First, separate PFC stripes are used to encode each step in the sequence. Thus, binding of phoneme identity and sequential order occurs in this model by using distinct neural substrates to represent the sequential information. This is entirely feasible because each stripe can represent all of the possible phonemes, given that they represent a closed class of items. Second, the storage of a

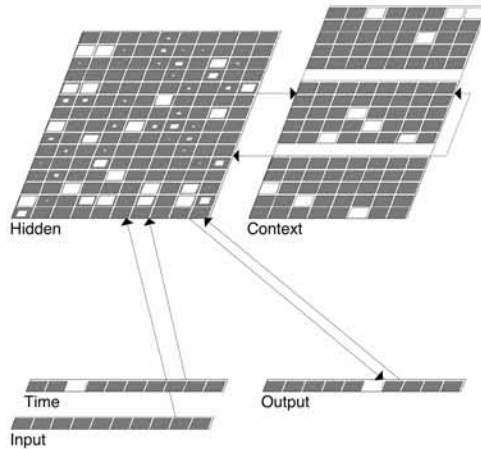

Figure 1: Phonological loop model. Ten different input symbols are possible at each time step (one unit out of ten activated in the Input layer). A sequence is encoded in one pass by presenting the Input together with the sequential location in the Time input layer for each step in the sequence. The simulated basal ganglia gating mechanism (implemented by fiat in script code) uses the time input to trigger intracellular maintenance currents in the corresponding stripe region of the context (PFC) layer (stripes are shown as the three separate groups of units within the Context layer; individual context units also had an excitatory self-connection for maintenance). Thus, the first stripe must learn to encode the first input, etc. Immediately after encoding, the network is then trained to produce the correct output in response to the time input, without any Input activation (the activation state shown is the network correctly recalling the third item in a sequence). The hidden layer must therefore learn to decode the context representations for this recall phase. Generalization testing involved presenting untrained sequences.

new sequence involves the basal ganglia gating mechanism triggering updates of the different PFC stripes in the appropriate order. We assume this can be learned over experience, and we are currently working on developing powerful learning mechanisms for adapting the basal ganglia gating mechanism in this way. This kind of gating control would also likely require some kind of temporal/sequential input that indicates the location within the sequence — such information might come from the cerebellum (e.g., Ivry, 1996).

In advance of having developed realistic and computationally powerful mechanisms for both the learning and the temporal/sequential control aspects of the model, we simply implemented these by fiat in the simulator. For the temporal signal indicating location within the sequence, we simply activated a different individual time unit for each point in the sequence (the Time input layer in Figure 1). This signal was then used by a simulated gating mechanism (implemented in script code in the simulator) to update the corresponding stripe in prefrontal cortex. Although the resulting model was therefore simplified, it nevertheless still had a challenging learning task to perform. Specifically, the stripe context layers had to learn to encode and maintain the current input value properly, and the Hidden layer had to be able to decode the context layer information as a function of the time input value. The model was implemented using the Leabra algorithm with standard parameters (O'Reilly, 1998; O'Reilly & Munakata, 2000).

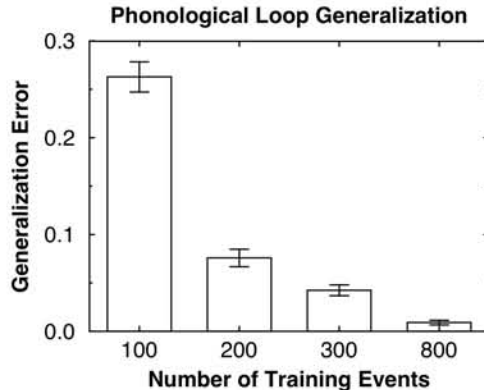

Figure 2: Generalization results for the phonological loop model as a function of number training patterns. Generalization is over 90% correct with training on less than 20% of the possible input patterns. $N = 5$.

## 3.1 Network Training

The network was trained as follows. Sequences (of length 3 for our initial work) were presented by sequentially activating an input "phoneme" and a corresponding sequential location input (in the Time input layer). We only used 10 different phonemes, each of which was encoded locally with a different unit in the Input layer. For example, the network could get Time = 0, Input = 2, then Time = 1, Input = 7, then Time = 2, Input = 3 to encode the sequence 2,7,3. During this encoding phase, the network was trained to activate the current Input on the Output layer, and the simulated gating function simply activated the intracellular maintenance currents for the units in the stripe in the Context (PFC) layer that corresponded to the Time input (i.e., stripe 0 for Time=0, etc). Then, the network was trained to recall this sequence, during which time no Input activation was present. The network received the sequence of Time inputs (0,1,2), and was trained to produce the corresponding Output for that location in the sequence (e.g., 2,7,3). The PFC context layers just maintained their activation states based on the intracellular ion currents activated during encoding (and recurrent activation) — once the network has been trained, the active PFC state represents the entire sequence.

## 3.2 Generalization Results

A critical test of the model is to determine whether it can perform systematically with novel sequences — only if it demonstrates this capacity can it serve as a mechanism for rapidly binding arbitrary information (such as the task demands studied by Miyake & Soto, in preparation). With 10 input phonemes and sequences of length three, there were 1,000 different sequences possible (we allowed phonemes to repeat). We trained on 100, 200, 300, and 800 of these sequences, and tested generalization on the remaining sequences. The generalization results are shown in Figure 2, which clearly shows that the network learned these sequences in a systematic manner and could transfer its training knowledge to novel sequences. Interestingly, there appears to be a critical transition between 100 and 200 training sequences — 100 sequences corresponds to each item within each slot being presented roughly 10 times, which appears to provide sufficient statistical information regarding the independence of individual slots.

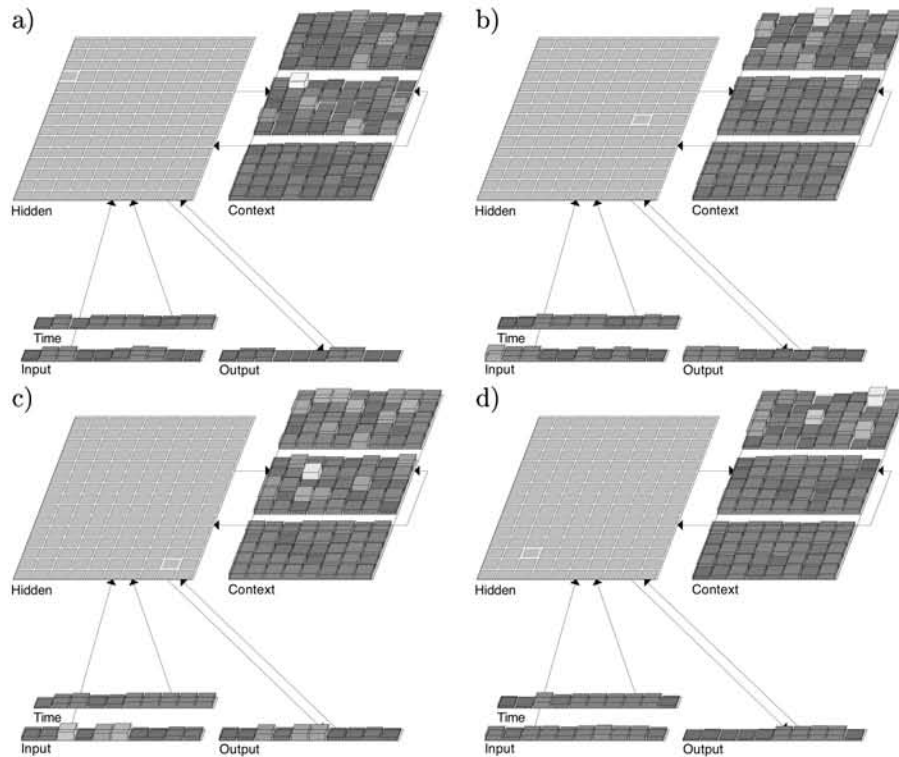

Figure 3: Hidden unit representations (values are weights into a hidden unit from all other layers). Unit in a) encodes the conjunction of a subset of input/output items at time 2. (b) encodes a different subset of items at time 2. (c) encodes items over times 2 and 3. (d) has no selectivity in the input, but does project to the output and likely participates in recall of items at time step 3.

## 3.3 Analysis of Representations

To understand how the hidden units encode and retrieve information in the maintained context layer in a systematic fashion that supports the good generalization observed, we examined the patterns of learned weights. Some representative examples are shown in Figure 3. Here, we see evidence of coarse-coded representations that encode a subset of items in either one time point in the sequence or a couple of time points. Also we found units that were more clearly associated with retrieval and not encoding. These types of representations are consistent with our other work showing how these kinds of representations can support good generalization (O'Reilly & Busby, 2002).

## 4  Discussion

We have presented a model of sequential encoding of phonemes, based on independently-motivated computational and biological considerations, focused on the neural substrates of the prefrontal cortex and basal ganglia (Frank et al., 2001). Viewed in more abstract, functional terms, however, our model is just another in a long line of computational models of how people might encode sequential order information. There are two classic models: (a) *associative chaining*, where the acti-

vation of a given item triggers the activation of the next item via associative links, and (b) *item-position association* models where items are associated with their sequential positions and recalled from position cues (e.g., Lee & Estes, 1977). The basic associative chaining model has been decisively ruled out based on error patterns (Henson, Norris, Page, & Baddeley, 1996), but modified versions of it may avoid these problems (e.g., Lewandowsky & Murdock, 1989). Probably the most accomplished current model, Burgess and Hitch (1999), is a version of the item-position association model with a competitive queuing mechanism where the most active item is output first and is then suppressed to allow other items to be output.

Compared to these existing models, our model is unique in not requiring fast associational links to encode items within the sequence. For example, the Burgess and Hitch (1999) model uses rapid weight changes to associate items with a context representation that functions much like the time input in our model. In contrast, items are maintained strictly via persistent activation in our model, and the basal-ganglia based gating mechanism provides a means of encoding items into separate neural slots that implicitly represent sequential order. Thus, the time inputs act independently on the basal ganglia, which then operates generically on whatever phoneme information is presently activated in the auditory input, obviating the need for specific item-context links.

The clear benefit of not requiring associational links is that it makes the model much more flexible and capable of generalization to novel sequences as we have demonstrated here (see O'Reilly & Munakata, 2000 for extended discussion of this general issue). Thus, we believe our model is uniquely well suited for explaining the role of the phonological loop in rapid binding of novel task information. Nevertheless, the present implementation of the model has numerous shortcomings and simplifications, and does not begin to approach the work of Burgess and Hitch (1999) in accounting for relevant psychological data. Thus, future work will be focused on remedying these limitations. One important issue that we plan to address is the interplay between the present model based on the prefrontal cortex and the binding that the hippocampus can provide — we suspect that the hippocampus will contribute item-position associations and their associated error patterns and other phenomena as discussed in Burgess and Hitch (1999).

### Acknowledgments

This work was supported by ONR grant N00014-00-1-0246 and NSF grant IBN-9873492. Rodolfo Soto died tragically at a relatively young age during the preparation of this manuscript — this work is dedicated to his memory.

## 5    References

Baddeley, A., Gathercole, S., & Papagno, C. (1998). The phonological loop as a language learning device. *Psychological Review, 105*, 158.

Baddeley, A. D. (1986). *Working memory.* New York: Oxford University Press.

Burgess, N., & Hitch, G. J. (1999). Memory for serial order: A network model of the phonological loop and its timing. *Psychological Review, 106*, 551–581.

Cohen, J. D., Braver, T. S., & O'Reilly, R. C. (1996). A computational approach to prefrontal cortex, cognitive control, and schizophrenia: Recent developments and current challenges. *Philosophical Transactions of the Royal Society (London) B, 351*, 1515–1527.

Cohen, J. D., & O'Reilly, R. C. (1996). A preliminary theory of the interactions between prefrontal cortex and hippocampus that contribute to planning and prospective memory. In M. Brandimonte, G. O. Einstein, & M. A. McDaniel (Eds.), *Prospective memory: Theory and applications* (pp. 267–296). Mahwah, New Jersey: Erlbaum.

Frank, M. J., Loughry, B., & O'Reilly, R. C. (2001). Interactions between the frontal cortex and basal ganglia in working memory: A computational model. *Cognitive, Affective, and Behavioral Neuroscience, 1*, 137–160.

Henson, R. N. A., Norris, D. G., Page, M. P. A., & Baddeley, A. D. (1996). Unclaimed memory: Error patterns rule out chaining models of immediate serial recall. *Quarterly Journal of Experimental Psychology: Human Experimental Psychology, 49(A)*, 80–115.

Ivry, R. (1996). The representation of temporal information in perception and motor control. *Current Opinion in Neurobiology, 6*, 851–857.

Lee, C. L., & Estes, W. K. (1977). Order and position in primary memory for letter strings. *Journal of Verbal Learning and Verbal Behavior, 16*, 395–418.

Levitt, J. B., Lewis, D. A., Yoshioka, T., & Lund, J. S. (1993). Topography of pyramidal neuron intrinsic connections in macaque monkey prefrontal cortex (areas 9 & 46). *Journal of Comparative Neurology, 338*, 360–376.

Lewandowsky, S., & Murdock, B. B. (1989). Memory for serial order. *Psychological Review, 96*, 25–57.

Mel, B. A., & Fiser, J. (2000). Minimizing binding errors using learned conjunctive features. *Neural Computation, 12*, 731–762.

Miller, E. K., & Cohen, J. D. (2001). An integrative theory of prefrontal cortex function. *Annual Review of Neuroscience, 24*, 167–202.

Miyake, A., & Soto, R. (in preparation). The role of the phonological loop in executive control.

O'Reilly, R. C. (1998). Six principles for biologically-based computational models of cortical cognition. *Trends in Cognitive Sciences, 2*(11), 455–462.

O'Reilly, R. C., Braver, T. S., & Cohen, J. D. (1999). A biologically based computational model of working memory. In A. Miyake, & P. Shah (Eds.), *Models of working memory: Mechanisms of active maintenance and executive control.* (pp. 375–411). New York: Cambridge University Press.

O'Reilly, R. C., & Busby, R. S. (2002). Generalizable relational binding from coarse-coded distributed representations. *Advances in Neural Information Processing Systems (NIPS), 2001.*

O'Reilly, R. C., Busby, R. S., & Soto, R. (in press). Three forms of binding and their neural substrates: Alternatives to temporal synchrony. In A. Cleeremans (Ed.), *The unity of consciousness: Binding, integration, and dissociation.* Oxford: Oxford University Press.

O'Reilly, R. C., & Munakata, Y. (2000). *Computational explorations in cognitive neuroscience: Understanding the mind by simulating the brain.* Cambridge, MA: MIT Press.

O'Reilly, R. C., & Rudy, J. W. (2001). Conjunctive representations in learning and memory: Principles of cortical and hippocampal function. *Psychological Review, 108*, 311–345.

Paulesu, E., Frith, C. D., & Frackowiak, R. S. J. (1993). The neural correlates of the verbal component of working memory. *Nature, 362*, 342–345.

Wickelgren, W. A. (1969). Context-sensitive coding, associative memory, and serial order in (speech) behavior. *Psychological Review, 76*, 1–15.
